# Sparse Instrumental Variables (SPIV) for Genome-Wide Studies

**Felix V. Agakov**
Public Health Sciences
University of Edinburgh
felixa@aivalley.com

**Paul McKeigue**
Public Health Sciences
University of Edinburgh
paul.mckeigue@ed.ac.uk

**Jon Krohn**
WTCHG, Oxford
jon.krohn@magd.ox.ac.uk

**Amos Storkey**
School of Informatics
University of Edinburgh
a.storkey@ed.ac.uk

## Abstract

This paper describes a probabilistic framework for studying associations between multiple genotypes, biomarkers, and phenotypic traits in the presence of noise and unobserved confounders for large genetic studies. The framework builds on sparse linear methods developed for regression and modified here for inferring causal structures of richer networks with latent variables. The method is motivated by the use of genotypes as "instruments" to infer causal associations between phenotypic biomarkers and outcomes, without making the common restrictive assumptions of instrumental variable methods. The method may be used for an effective screening of potentially interesting genotype-phenotype and biomarker-phenotype associations in genome-wide studies, which may have important implications for validating biomarkers as possible proxy endpoints for early-stage clinical trials. Where the biomarkers are gene transcripts, the method can be used for fine mapping of quantitative trait loci (QTLs) detected in genetic linkage studies. The method is applied for examining effects of gene transcript levels in the liver on plasma HDL cholesterol levels for a sample of sequenced mice from a heterogeneous stock, with $\sim 10^5$ genetic instruments and $\sim 47 \times 10^3$ gene transcripts.

## 1 Introduction

A problem common to both epidemiology and to systems biology is to infer causal relationships between phenotypic measurements (biomarkers) and disease outcomes or quantitative traits. The problem is complicated by the fact that in large bio-medical studies, the number of possible genetic and environmental causes is very large, which makes it implausible to conduct exhaustive interventional experiments. Moreover, it is generally impossible to remove the confounding bias due to unmeasured latent variables which influence associations between biomarkers and outcomes. Also, in situations when the biomarkers are mRNA transcript levels, the measurements are known to be quite noisy; additionally, the number of unique candidate causes may exceed the number of observations by several orders of magnitude (the $p \gg n$ problem). A fundamentally important practical task is to reduce the number of possible causes of a trait to a much more manageable subset of candidates for controlled interventions. Developing an efficient framework for addressing this problem may be fundamental for overcoming bottlenecks in drug development, with possible applications in the validation of biomarkers as causal risk factors, or developing proxies for clinical trials.

Whether or not causation may be inferred from observational data has been a matter of philosophical debate. Pearl [28] argues that causal assumptions cannot be verified unless one makes a recourse

to experimental control, and that there is nothing in the probability distribution $p(x, y)$ which can tell whether a change in $x$ may have an effect on $y$. Traditional discussions of causality are largely focused on the question of identifiability, i.e. determining sets of graph-theoretic conditions when a post-intervention distribution $p(y|do(x))$ may be uniquely determined from a pre-intervention distribution $p(y, x, z)$ [27, 4, 32]. If the causal effects are shown to be identifiable, their magnitudes can be obtained by statistical estimation, which for common models often reduces to solving systems of linear equations. In contrast, from the Bayesian perspective, the causality detection problem may be viewed as that of model selection, where a model $\mathcal{M}_{x \to y}$ is compared with $\mathcal{M}_{y \to x}$. The problem is complicated by the likelihood-equivalence, where for each setting of parameters of one model there may exist a setting of parameters of the other giving rise to the identical likelihoods. However, unless the priors are chosen in such a way that $\mathcal{M}_{x \to y}$ and $\mathcal{M}_{y \to x}$ also have identical posteriors, it may be possible to infer the direction of the arrow. The view that the priors of likelihood-equivalent models do not need to be set to ensure the equivalence of the posteriors is in contrast to e.g. [12] (and references therein), but has been defended by MacKay (see [21], Section 35).

In this paper we are leaving aside debates about the nature of causality and focus instead on identifying a set of candidate causes for a large partially observed under-determined genetic problem. The approach builds on the instrumental variable methods that were historically used in epidemiological studies, and on approximate Bayesian inference in sparse linear latent variable models. Specific modeling hypotheses are tested by comparing approximate marginal likelihoods of the corresponding direct, reverse, and pleiotropic models with and without latent confounders, where we follow [21] in allowing for flexible priors. The approach is largely motivated by the observation that independent variables do not establish a causal relation, while strong unconfounded direct dependencies retained in the posterior modes even under large sparseness-inducing penalties may indicate potential causality and suggest candidates for further controlled experiments.

## 2 Previous work

Inference of causal direction of $x$ on $y$ is to some extent simplified if we assume existence of an auxiliary variable $g$, such that $g$'s effect on $x$ may only be causal, and $g$'s effect on $y$ may only be through $x$. The idea is exploited in *instrumental variable* methods [3, 2, 29] which typically deal with low-dimensional linear models, where the strength of the causal effect may be estimated as $w_{x \to y} = \text{cov}(g, y)/\text{cov}(g, x)$. Note also that the hypothesized cause-outcome models such as $\mathcal{M}_{g \to x \to y}$ and $\mathcal{M}_{g \to y \to x}$ are no longer Markov-equivalent, i.e. it may be possible to select an appropriate model via likelihood-based tests. Selecting a plausible *instrument* $g$ may be difficult in some domains; however, in genetic studies it may be possible to exploit as an instrument a measure of genotypic variation. In quantitative genetics, such applications of instrumental variable methods have been termed *Mendelian randomization* [15, 34]. In accordance with the requirements of the classic instrumental variable methods, it is assumed that effects of the genetic instrument $g$ on the biomarker $x$ are unconfounded, and that effects of the instrument on the outcome $y$ are mediated only through the biomarker (i.e. there is *no pleiotropy*) [17, 35]. The former assumption is grounded in the laws of Mendelian genetics and is satisfied as long as population stratification has been adequately controlled. However, the assumption of no hidden pleiotropy severely restricts the application of this approach, as most genotypic effects on complex traits are not sufficiently well understood to exclude pleiotropy as a possible explanation of an association. Thus the classical instrumental variable argument is limited to biomarkers for which suitable non-pleiotropic instruments exist, and cannot be easily extended to exploit studies with multiple biomarkers and genome-wide data.

A more general approach to exploiting genotypic variation to infer causal relationships between gene transcript levels and quantitative traits has been developed by Schadt et. al. [30] and subsequently extended (see e.g. [5]). They relax the assumption of no pleiotropy, but instead compare models with and without pleiotropy by computing standard likelihood-based scores. After filtering to select a set of gene transcripts $\{x_j\}$ that are associated with the trait $y$, and loci $\{g_i\}$ at which genotypes have effects on transcript levels $x_j$, each possible triad of marker locus $g_i$, transcript $x_j$ and trait $y$ is evaluated to compare three possible models: causal effect of transcript on trait, reverse causation, and a pleiotropic model (see Figure 1 *left, (i)–(iii)*). The support for these three models is compared by a measure of model fit penalized by complexity: either Akaike's Information Criterion (AIC) [30], or the Bayesian Information Criterion (BIC) [5]. Schadt et. al. [30] denote this procedure as the "likelihood-based causality model selection" (LCMS) approach. While the LCMS

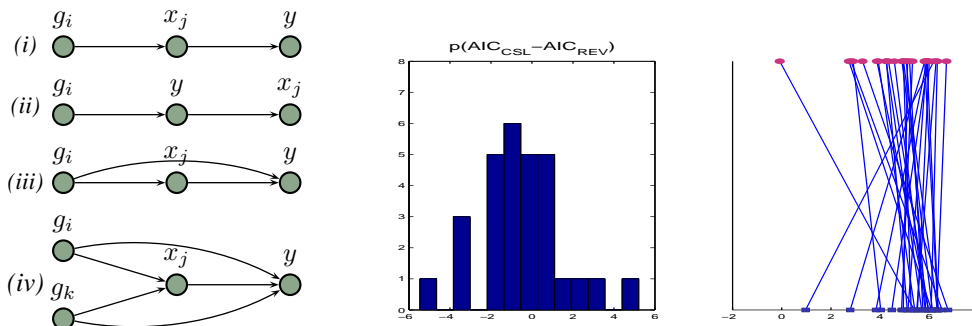

Figure 1: *Left: (i–iii):* Causal, reverse, and pleiotropic models of the LCMS approach [30]; (iv): pleiotropic model with two genetic instruments. *Center:* Possible arbitrariness of LCMS inference. The histogram shows the difference of the AIC scores for the causal and reverse models for a fixed biomarker and outcome, and various choices of loci from predictive regions. *Right:* AIC scores of the causal (top) and reverse (bottom) models for each choice of instrument $g_i$ (the straight lines link the scores for a fixed choice of $g_i$). Scores were centered relative to those of the pleiotropic model. Biomarker and outcome are liver expressions of *Cyp27b1* and plasma HDL measurements for heterogeneous mice. Based on the choice of $g_i$, either causal or reverse explanations are favored.

and related methods [30, 5] relax the assumption of no hidden pleiotropy of the classic Mendelian randomization method, they have three key limitations. First, effects of loci and biomarkers on outcomes are not modeled jointly, so widely varying inferences are possible depending on the choice of the triads $\{g_i, x_j, y\}$. Figure 1 *center, right* compares differences in the AIC scores for the causal and reverse models constructed for a fixed biomarker and outcome, and for various choices of the genetic instruments from the predictive region. Depending on the choice of instrument $g_i$, either causal or reverse explanations are favored. A second key limitation is that the LCMS method does not allow for dependencies between multiple biomarkers, measurement noise, or latent variables (such as unobserved confounders of the biomarker-outcome associations). Thus, for instance, without allowance for noise in the biomarker measurements, non-zero conditional mutual information $I(g_i, y | x_j)$ will be interpreted as evidence of pleiotropy or reverse causation even when the relation between the underlying biomarker and outcome is causal. Also, the method is not Bayesian (the BIC score is only a crude approximation to the Bayesian procedure for model selection).

One extension of the classic instrumental variable methods has been proposed by [4], who described graph-theoretic conditions which need to be satisfied in order for parameters of edges $x_i \rightarrow y$ to be identifiable by solving a system of linear equations; however, they focus on the identifiability problem rather than on addressing a large practical under-determined task with latent variables. For example, their method does not allow for an easy integration of unmeasured confounders with unknown correlations with the intermediate and outcome variables. Another approach to modeling joint effects of genetic loci and biomarkers (gene expressions) was described by [41]. They modeled the expression measurements as three ordered levels, and used a biased greedy search over model structures from multiple starting points, to find models with high BIC scores. Though applicable for large-scale studies, the approach does not allow for measurement noise or latent variables (and looses information by using categorical measurements). The vast majority of other recent model selection and structure learning methods from machine learning literature are also either not easily extended to include latent confounders (e.g. [16], [19], [22]), or applicable only for dealing with relatively low-dimensional problems with abundant data (e.g. [33] and references therein).

## 3 Methods

To address the problem of causal discovery in large bio-medical studies, we need a unified framework for modeling relations between genotypes, biomarkers, and outcomes that is computationally tractable to handle a large number of variables. Our approach extends LCMS and the instrumental variable methods by the joint modeling of effects of genetic loci and biomarkers, and by allowing for both pleiotropic genotypic effects and latent variables that generate couplings between biomarkers and confound the biomarker-outcome associations. It relies on Bayesian modeling of linear associations between the modeled variables, with sparseness-inducing priors on the linear weights. The

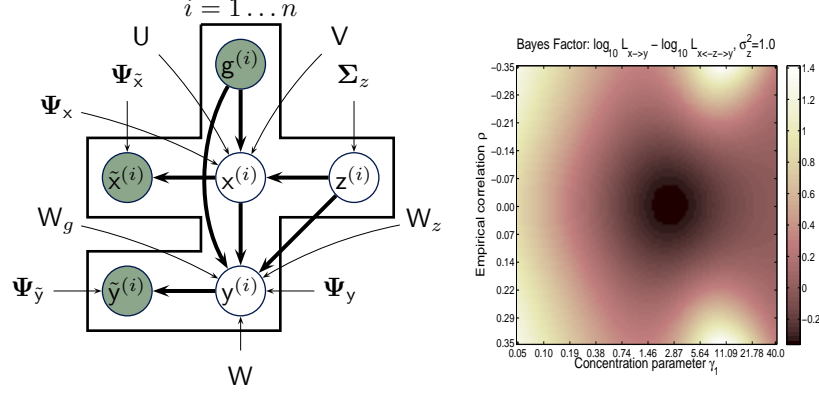

Figure 2: *Left:* SPIV structure. Filled/clear nodes correspond to observed/ latent variables. *Right:* log Bayes factor of $\mathcal{M}_{x \leftarrow z \rightarrow y}$ and $\mathcal{M}_{x \rightarrow y}$ as a function of empirical correlations $\rho$ and $\gamma_1$ for $n = 100$ observations, $\sigma_z^2 = \sigma_x^2 = \sigma_y^2 = 1$, $|x| = |y| = |z| = 1$ and $\gamma_2 = 0$, on the $\log_{10}$ scale. For intermediate $\gamma_1$'s and high empirical correlations, there is a strong preference for the causal model.

Bayesian framework allows prior biological information to be included if available: for instance, cis-acting genotypic effects on transcript levels are likely to be stronger and less pleiotropic than trans-acting effects on transcript levels. It also offers a rigorous approach to model comparison, and is particularly attractive for addressing under-determined genetics problems ($p \gg n$). The method builds on automatic relevance determination approaches (e.g. [20], [25], [37]) and adaptive shrinkage (e.g. [36], [8], [42]). Here it is used in the context of sparse multi-factor instrumental variable analysis in the presence of unobserved confounders, pleiotropy, and noise.

**Model Parameterization**

Our *sparse instrumental variables* model (SPIV) is specified with four classes of variables: genotypic and environmental covariates $g \in \mathbb{R}^{|g|}$, phenotypic biomarkers $x \in \mathbb{R}^{|x|}$, outcomes $y \in \mathbb{R}^{|y|}$, and latent factors $z_1, \ldots, z_{|z|}$. The dimensionality of the latent factors $|z|$ is fixed at a moderately high value (extraneous dimensions will tend to be pruned under the sparse prior). The latent factors $z$ play two major roles: they represent the shared structure between groups of biomarkers, and confound biomarker-outcome associations. The biomarkers $x$ and outcomes $y$ are specified as hidden variables inferred from noisy observations $\tilde{x} \in \mathbb{R}^{|\tilde{x}|}$ and $\tilde{y} \in \mathbb{R}^{|\tilde{y}|}$ (note that $|\tilde{x}| = |x|$, $|\tilde{y}| = |y|$). The effects of genotype on biomarkers and outcome are assumed to be unconfounded. Pleiotropic effects of genotype (effects on outcome that are not mediated through the phenotypic biomarkers) are accounted for by an explicit parameterization of $p(y|g, x, z)$. Graphical representation of the model is shown on Figure 2 (*left*). It is clear that the SPIV structure extends that of the instrumental variable methods [2, 3, 29] by allowing for the pleiotropic links, and also extends the pleiotropic model of Schadt et. al. [30] (Figure 1 *left (iii)*) by allowing for multiple instruments and latent variables.

All the likelihood terms of $p(x, \tilde{x}, y, \tilde{y}, z|g)$ are linear Gaussians with diagonal covariances

$$x = U^T g + V^T z + e_x, \quad y = W^T x + W_z^T z + W_g^T g + e_y, \quad \tilde{x} = Ax + e_{\tilde{x}}, \tag{1}$$

and $\tilde{y} = y + e_{\tilde{y}}$, where $e_x \sim \mathcal{N}(0, \boldsymbol{\Psi}_y)$, $e_y \sim \mathcal{N}(0, \boldsymbol{\Psi}_y)$, $e_{\tilde{y}} \sim \mathcal{N}(0, \boldsymbol{\Psi}_{\tilde{y}})$, $e_{\tilde{x}} \sim \mathcal{N}(0, \boldsymbol{\Psi}_{\tilde{x}})$, $z \sim \mathcal{N}(0, \boldsymbol{\Psi}_z)$, $W \in \mathbb{R}^{|x| \times |y|}$, $W_z \in \mathbb{R}^{|z| \times |y|}$, $W_g \in \mathbb{R}^{|g| \times |y|}$, $V \in \mathbb{R}^{|z| \times |x|}$, $U \in \mathbb{R}^{|g| \times |x|}$ are regression coefficients (factor loadings) – for clarity, we assume the data is centered. $A \in \mathbb{R}^{|x| \times |x|}$ has a banded structure (accounting for possible couplings of the neighboring microarray measurements).

**Prior Distribution**

All model parameters are specified as random variables with prior distributions. For computational convenience, the variance components of the diagonal covariances $\boldsymbol{\Psi}_y$, $\boldsymbol{\Psi}_{\tilde{y}}$, etc. are specified with inverse Gamma priors $\Gamma^{-1}(a_i, b_i)$, with hyperparameters $a_i$ and $b_i$ fixed at values motivating the prior beliefs about the projection noise (often available to lab technicians collecting trait or biomarker measurements). One way to view the latent confounders $z$ is as missing genotypes or environmental covariates, so that prior variances of the latent factors are peaked at values representative of the empirical variances of the instruments $g$. Empirically, the choice of priors on the variance components appears to be relatively unimportant, and other choices may be considered [9].

The considered choice of a sparseness-inducing prior on parameters $\mathsf{W}$, $\mathsf{W}_z$, $\mathsf{W}_g$, etc. is a product of zero-mean Laplace and zero-mean normal distributions

$$p(\mathsf{w}) \propto \prod_{i=1}^{|\mathsf{w}|} \mathcal{L}_{w_i}(0, \gamma_1) \mathcal{N}_{w_i}(0, \gamma_2), \tag{2}$$

$\mathcal{L}_{w_i}(0, \gamma_1) \propto \exp\{-\gamma_1|w_i|\}$, and $\mathcal{N}_{w_i}(0, \gamma_2) \propto \exp\{-\gamma_2 w_i^2\}$. Due to the heavy tails of the Laplacian $\mathcal{L}_{w_i}$, the prior $p(\mathsf{w})$ is flexible enough to capture large associations even if they are rare. Higher values of $\gamma_1$ give a stronger tendency to shrink irrelevant weights to zero. It is possible to set different $\gamma_1$ parameters for different linear weights (e.g. for the cis- and trans-acting effects); however, for clarity of this presentation we shall only use a global parameter $\gamma_1$. The isotropic Gaussian component with the inverse variance $\gamma_2$ contributes to the grouping effect (see [42], Theorem 1). The considered family of priors (2) induces better consistency properties [40] than the commonly used Laplacians [36, 9, 39, 26, 31]. It has also been shown [14] that important associations between variables may be recovered even for severely under-determined problems ($p \gg n$) common in genetics. The SPIV model with $p(\mathsf{w})$ defined as in (2) generalizes LASSO and elastic net regression [36, 42]. As a special case, it also includes sparse conditional factor analysis. Other sparse priors on the weights, such as Student-$t$, "spike-and-slab", or inducing $L_{q<1}$ penalties tend to result in less tractable posteriors even for linear regression [10, 37, 8], which also motivates the choice (2).

Some additional intuition of the influence of the sparse prior on the causal inference may be gained by numerically comparing the marginal likelihoods of the Markov-equivalent models with and without confounders $\mathcal{M}_{x \leftarrow z \rightarrow y}$, $\mathcal{M}_{x \rightarrow y}$. (Comparison of these models is of particular importance in epidemiology, because while the temporal data may often be available for distinguishing direct and reverse models $\mathcal{M}_{x \rightarrow y}$ and $\mathcal{M}_{y \rightarrow x}$, it is generally difficult to ensure that there is no confounding). Figure 2 shows that when the empirical correlations are strong and $\gamma_1$ is at intermediate levels, there is a strong preference for a causal model. This is because the alternative model with the confounders will have more parameters, and the weights will need to be larger (and therefore more strongly penalized by the prior) in order to lead to the same likelihood (note that for $\mathrm{var}(x) = \mathrm{var}(y) = 1$, the likelihood-equivalence is achieved for $w = vw_z$, $|w| \leq 1$). Larger values of $\gamma_1$ will tend to strongly penalize all the weights, which makes the models largely indistinguishable. Also, as the number of genetic instruments grows, evidence in favor of the causal or pleiotropic model will be less dependent upon the priors on model parameters. For instance, with two genotypic variables that perturb a single transcript, the causal model has three adjustable parameters, but the pleiotropic model has five (see Figure 1 *left, (iv)*). Where several genotypic variables perturb a single transcript and the causal model fits the data nearly as well as the pleiotropic model, the causal model will have higher marginal likelihood under almost any plausible prior, because the slightly better fit of the pleiotropic model will be outweighed by the penalty imposed by several extra adjustable parameters.

**Inference**

While the choice of prior (2) encourages sparse solutions, it makes exact inference of the posterior parameters $p(\theta|\mathcal{D})$ analytically intractable. The most efficient approach is based on the maximum-a-posteriori (*MAP*) treatment ([36], [9]), which reduces to solving the optimization problem

$$\theta_{MAP} = \arg\max_{\theta} \left\{ \log p\left(\{\tilde{\mathsf{y}}\}, \{\tilde{\mathsf{x}}\}|\{\mathsf{g}\}, \theta\right) + \log p(\theta) \right\} \tag{3}$$

for the joint parameters $\theta$, where the latent variables have been integrated out. Note that the MAP solution for SPIV may also be easily derived for the semi-supervised case where the biomarker and outcome vectors are only partially observed. Compared to other approximations of inference in sparse linear models based e.g. on sampling or expectation propagation [26, 31], the MAP approximation allows for an efficient handling of very large networks with multiple instruments and biomarkers, and makes it straightforward to incorporate latent confounders. Depending on the choice of the global sparseness and grouping hyperparameters $\gamma_1, \gamma_2$, the obtained solutions for the weights will tend to be sparse, which is also in contrast to the full inference methods. In high dimensions in particular, the parsimony induced by the point-estimates will facilitate structure discovery and interpretations of the findings.

One way to optimize (3) is by an EM-like algorithm. For example, the fixed-point update for $\mathsf{u}_i \in \mathbb{R}^{|\mathsf{g}|}$ linking biomarker $x_i$ with the vector of instruments $\mathsf{g}$ is easily expressed as

$$\mathsf{u}_i^{(t)} = \left( \mathsf{G}^T \mathsf{G} + \sigma_{x_i}^2 \left( \gamma_1 \acute{\mathsf{U}}_i^{(t-1)} + \gamma_2 \mathsf{I}_{|\mathsf{g}|} \right) \right)^{-1} \left( \mathsf{G}^T \langle \mathsf{x}_i \rangle - \mathsf{G}^T \langle \mathsf{Z} \rangle \mathsf{v}_i \right), \tag{4}$$

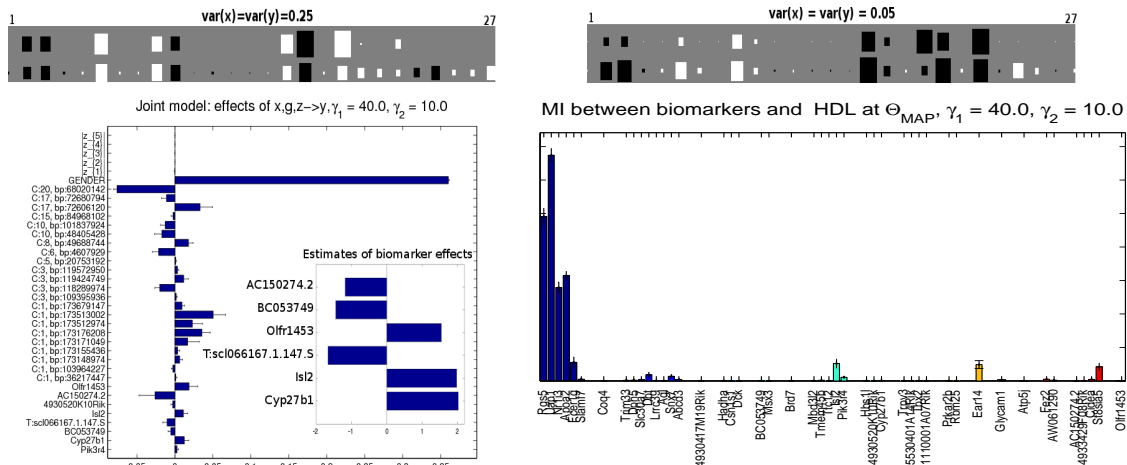

Figure 3: *Top:* SPIV for artificial datasets. Left/right plots show typical applications for the high and low observation noise ($\sigma_{\tilde{x}}^2 = 0.25$ and $\sigma_{\tilde{x}}^2 = 0.05$ respectively). Top and bottom rows of each Hinton diagram correspond to the ground truth and the MAP weights $\mathsf{U}$ (1–18), $\mathsf{W}$ (19–21), $\mathsf{W}_g$ (22–27). *Bottom:* SPIV for a genome-wide study of causal effects on HDL in heterogeneous stock mice. Left/right plots show maximum a-posteriori weights $\theta_{MAP}$ and the mutual information $I(x_i, y|\mathsf{e})$ between the unobserved biomarkers and outcome evaluated from the model at $\theta_{MAP}$, under the joint Gaussian assumption. A cluster of pleiotropic links on chromosome 1 at about 173 MBP is consistent with biology. The biomarker with the strongest unconfounded effect on HDL is *Cyp27b1*. Transcripts that are most predictive of HDL through their links with pleiotropic genetic markers on chrom 1 are *Uap1, Rgs5, Apoa2*, and *Nr1i3*. Parameters $\gamma_{1,2}$ have been obtained by cross-validation.

where $\mathsf{G} \in \mathbb{R}^{n \times |\mathsf{g}|}$ is the design matrix, $(\acute{\mathsf{U}}_i)_{kl} = \delta_{kl}/|u_{ki}| \ \forall k, l \in [1, |\mathsf{g}|] \cap \mathbb{Z}$, $x_i \in \mathbb{R}^n$, $\mathsf{Z} \in \mathbb{R}^{n \times |\mathsf{z}|}$, $\mathsf{v}_i \in \mathbb{R}^{|\mathsf{z}|}$, and $\sigma_{x_i}^2 = (\Psi_x)_{ii}$. The expectations $\langle . \rangle$ are computed with respect to $p(.|\{\tilde{x}\}, \{\tilde{y}\}, \{\mathsf{g}\})$, which for (1) are easily expressed in the closed form. The rest is expressed analogously, and extensions to the partially observed cases are straight-forward. Faster (although more heuristic) alternatives may be used for speeding up the M-step (e.g. [7]). The hyperparameters may be set by cross-validation, marginalized out by specifying a hyper-prior, or set heuristically based on the expected number of links to be retained in the posterior mode. Once a sparse representation is produced by pruning irrelevant dimensions, more computationally-intensive inference methods for the full posterior (such as expectation propagation or MCMC) may be used in the resulting lower-dimensional model if needed. After fitting SPIV to data, formal hypotheses tests were performed by comparing the marginal likelihoods of the specific models for the retained instruments, biomarkers, and target outcomes. These were evaluated by the Laplace approximation at $\theta_{MAP}$ (e.g. [20]).

## 4 Results

**Artificial data:** We applied SPIV to several simulated datasets, and compared specific modeling hypotheses for the biomarkers retained in the posterior modes. The structures were consistent with the generic SPIV model, with all non-zero weights sampled from $\mathcal{N}(0, 1)$. Figure 3 (top) shows typical results for the high/low observation noise ($\forall i, \sigma_{\tilde{x}_i}^2 = \sigma_{\tilde{y}}^2 = 0.25/0.05$). Note excellent sign-consistency of the results for the more important factors. Separate simulations showed robustness under multiple EM runs and under- or over-estimation of the true number of confounders. Subsequent testing of the specific modeling hypotheses for the most important factors resulted in the correct discrimination of causal and confounded associations in $\approx 86\%$ of cases.

**Genome-wide study of HDL cholesterol in mice:** To demonstrate our method for a large-scale practical application, we examined effects of gene transcript levels in the liver on plasma high-density lipoprotein (HDL) cholesterol levels for a mice from a heterogeneous stock. The genetic factors influencing HDL in mice have been well explored in biology e.g. by Valdar et. al. [38]. The gene expression data was collected and preprocessed by [13], who have kindly agreed to share a part of their data. Breeding pairs for the stock were obtained at 50 generations after the stock

foundation. At each of the 12500 marker loci, genotypes were described by 8-D vectors of expected founder ancestry proportions inferred from the raw marker genotypes by an HMM-based reconstruction method [23]. Mouse-specific covariates included age and sex, which were used to augment the set of genetic instruments. The full set of phenotypic biomarkers consisted of 47429 transcript levels, appropriately transformed and cleaned. Available data included 260 animals. Before applying our method, we decreased the dimensionality of the genetic features and RNA expressions by using a combination of seven feature (subset) selection methods, based on applications of filters, greedy (step-wise) regression, sequential approximations of the mutual information between the retained set and the outcome of interest, and applications of regression methods with LASSO and elastic net (EN) shrinkage priors for the genotypes g, observed biomarkers $\tilde{x}$, and observed HDL measurements $\tilde{y}$. For the LASSO and EN methods, global hyper-parameters were obtained by 10-fold cross-validation. Note that feature selection is unavoidable for genome-wide studies using gene expressions as biomarkers. Indeed, the considered case of $\sim O(10^5)$ instruments and 47K biomarkers would give rise to $\gtrsim O(10^9)$ interaction weights, which is expensive to analyze or even keep in memory. After applying subset selection methods, SPIV was typically applied to subsets of data with $\sim O(10^5)$ loci-biomarker interactions.

The results of the SPIV analysis of this dataset are shown on Figure 3 (bottom). The *bottom left* plot shows maximum a-posteriori weights $\theta_{MAP}$ computed by running the EM-like optimization procedure to convergence from 20 random initializations. For a model with latent variables and about $30,000$ weights, each run took approximately 10 minutes of execution time (only weakly optimized Matlab code, simple desktop). The parameters $\gamma_{1,2}$ were obtained by 10-fold CV. Note that only a fraction of the variables remains in the posterior. In this case and for the considered sparseness-inducing priors, no hidden confounders appear to have strong effects on the outcome in the posterior[1]. The spikes of the pleiotropic activations in sex chromosome 20 and around chromosome 1 are consistent with the biological knowledge [38]. The biomarker with the strongest direct effect on HDL (computed as the mean MAP weight $w_i : x_i \to y$ divided by its standard deviation over multiple runs, where each mean weight exceeds a threshold) is the expression of *Cyp27b1* (gene responsible for vitamin D metabolism). Knockout of the *Cyp27b1* gene in mice has been shown to alter body fat stores [24], which might be expected to affect HDL cholesterol levels. Recently it has also been shown that quantitative trait locus for circulating vitamin D levels in humans includes a gene that codes for the enzyme that synthesizes cholesterol [1]. A subsequent comparison of 18 specific reverse, pleiotropic, and causal models for *Cyp27b1*, HDL, and the whole vector of retained genetic instruments (known to be causal by definition) showed a slightly stronger evidence in favor of the reverse hypothesis without latent confounders (with the ratio of Laplace approximations of the marginal likelihoods of reverse *vs* causal models of $\approx 1.95 \pm 0.27$). This is in contrast to the LCMS where the results are strongly affected by the choice of an instrument (Figure 1 *right* shows the results for *Cyp27b1*, HDL, and the same choice of instruments).

To demonstrate an application to gene fine-mapping studies, Figure 3 (*bottom right*) shows the approximate mutual information $I(x_i, y|\mathsf{e} = \{age, sex\})$ between the underlying biomarkers and unobserved HDL levels expressed from the model at $\theta_{MAP}$. The mutual information takes into account not only the strength of the direct effect of $x_i$ on $y$, but also associations with the pleiotropic instruments, strengths of the pleiotropic effects, and dependencies between the instruments. Under the as-if Gaussian assumption, $I(x_i, y_j|\theta_{MAP}) = \log(\sigma_{y_j}^2 \sigma_{x_i}^2) - \log(\sigma_{y_j}^2 \sigma_{x_i}^2 - \sigma_{y_j x_i}^4)$, where

$$\sigma_{y_j}^2 = \|\mathbf{\Sigma}_{gg}^{1/2}(\mathsf{U}\mathsf{w}_j + \mathsf{w}_{g_j})\|^2 + \|\mathbf{\Psi}_{\mathsf{z}}^{1/2}(\mathsf{V}\mathsf{w}_j + \mathsf{w}_{z_j})\|^2 + \mathsf{w}_j^T \mathbf{\Psi}_{\mathsf{x}}\mathsf{w}_j + \Psi_{y_j}, \tag{5}$$

with the rest expressed analogously. Here $\mathbf{\Sigma}_{gg} \in \mathbb{R}^{|\mathsf{g}| \times |\mathsf{g}|}$ is the empirical covariance of the instruments, $\mathsf{w}_j \in \mathbb{R}^{|\mathsf{x}|}$, $\mathsf{w}_{z_j} \in \mathbb{R}^{|\mathsf{z}|}$, and $\mathsf{w}_{g_j} \in \mathbb{R}^{|\mathsf{g}|}$ are the MAP weights of the couplings of $y_j$ with the biomarkers, confounders, and genetic instruments respectively. When the outcome is HDL, the majority of predictive transcripts are fine-mapped to a small region on chromosome 1 which includes *Uap1, Rgs5, Apoa2*, and *Nr1i3*. The informativeness of these genes about the HDL cholesterol cannot be inferred simply from correlations between the measured gene expression and HDL levels; for example, when ranked in accordance to $\rho^2(\tilde{x}_i, \tilde{y}|age, sex)$, the top 4 genes have the rankings

of 838, 961, 6284, and 65 respectively. The findings are also biologically plausible and consistent with high-profile biological literature (with associations between *Apoa2* and HDL described in [38], and strong links of *Rgs5* to a genomic region strongly associated with metabolic traits discussed in [5], while *Nr1i3* and *Uap1* are their neighbors on chromosome 1 within $\sim 1Mbp$). Note that the couplings are via the links with the pleiotropic genetic markers on chromosome 1. Adjusting for sex and age prior to performing feature selection and inference did not significantly change the results.

The results reported here appear to be stable for different choices of feature selection methods, data adjustments, and algorithm runs. We note, however, that different results may potentially be obtained based on the choice of animal populations and/or processing of the biomarker (gene expression) measurements. Details of the data collection, microarray preprocessing, and feature selection, along with the detailed findings for other biomarkers and phenotypic outcomes will be made available online. Definitive confirmation of these relationships would require gene knock-out experiments.

## 5  Discussion and extensions

In large-scale genetic and bio-medical studies, we are facing a practical task of reducing a huge set of candidate causes of complex traits to a more manageable subset of candidates where experimental control (such as gene knockout experiments or biomarker alternations) may be performed. SPIV performs the screening of interesting biomarker-phenotype and genotype-biomarker-phenotype associations by exploiting the maximum-a-posteriori inference in a sparse linear latent variable model. Additional screening is performed by comparing approximate marginal likelihoods of specific modeling hypotheses, including direct, reverse, and pleiotropic models with and without confounders, which (under the assumption of no "prior equivalence") may serve as an additional test of possible causation [21]. Intuitively, the approach is motivated by the observation that while independence of variables implies that they are not in a causal relation, a preference for an unconfounded causal model may indicate possible causality and require further controlled experiments.

Technically, SPIV may be viewed as an extension of LASSO and elastic net regression which allows for latent variables and pleiotropic dependencies. While being particularly attractive for genetic studies, SPIV or its modifications may potentially be applied for addressing more general structure learning tasks. For example, when applied iteratively, SPIV may be used to guide search over richer model structures (where a greedy search over parent nodes is replaced by a continuous optimization problem which combines subset selection and regression in the presence of latent variables), which may be used for structure learning problems. Other extensions of the framework could involve hybrid (discrete- and real-valued) outcomes with nonlinear/nongaussian likelihoods. Also, as mentioned earlier, once sparse representations are produced by the MAP inference, it may be possible to utilize more accurate approximations of the inference applicable for the induced sparse structures [6]. Also note that sparse priors on the linear weights tend to give rise to sparse covariance matrices. A potentially interesting alternative may involve a direct estimation of conditional precision matrices with a sparse group penalty. While SPIV attempts to focus the attention on *important* biomarkers establishing strong direct associations with the phenotypes, modeling of the precisions may be used for filtering out *unimportant* factors (conditionally) independent of the outcome variables. Our future work will involve a direct estimation of the sparse conditional precision matrix $\Sigma^{-1}_{xyz|g}$ of the biomarkers, outcomes, and unmeasured confounders (given the instruments), through latent variable extensions of the recently proposed graphical LASSO and related methods [11, 18].

The key purpose of this paper is to draw attention of the machine learning community to the problem of inferring causal relationships between phenotypic measurements and complex traits (disease risks), which may have tremendous implications in epidemiology and systems biology. Our specific approach to the problem is inspired by the ideas of instrumental variable analysis commonly used in epidemiological studies, which we have extended to properly address situations when the genetic variables may be direct causes of the hypothesized outcomes. The sparse instrumental variable framework (SPIV) overcomes limitations of the likelihood-based LCMS methods often used by geneticists, by modeling joint effects of genetic loci and biomarkers in the presence of noise and latent variables. The approach is tractable enough to be used in genetic studies with tens of thousands of variables. It may be used for identifying specific genes associated with phenotypic outcomes, and may have wide applications in identification of biomarkers as possible targets for interventions, or as proxy endpoints for early-stage clinical trials.

## Footnotes

[1]No confounder effects in the posterior mode for the considered $\gamma_{1,2}$ is specific to the considered mouse HDL dataset, which shows relatively strong correlations between the measured biomarkers and the outcome. An application of SPIV to proprietary human data for a study of effects of vitamins and calcium levels on colorectal cancer (which we are not yet allowed to publish) showed very strong effects of the latent confounders.

# References

[1] J. Ahn, K. Yu, and R. Stolzenberg-Solomon et. al. Genome-wide association study of circulating vitamin D levels. *Human Molecular Genetics*, 2010. Epub ahead of print.

[2] J. D. Angrist, G. W. Imbens, and D. B. Rubin. Identification of causal effects using instrumental variables (with discussion). *J. of the Am. Stat. Assoc.*, 91:444–455, 1996.

[3] R. J. Bowden and D. A. Turkington. *Instrumental Variables*. Cambridge Uni Press, 1984.

[4] C. Brito and J. Pearl. Generalized instrumental variables. In *UAI*, 2002.

[5] Y. Chen, J. Zhu, and P. Y. Lum et. al. Variations in DNA elucidate molecular networks that cause disease. *Nature*, 452:429–435, 2008.

[6] B. Cseke and T. Heskes. Improving posterior marginal approximations in latent Gaussian models. In *AISTATS*, 2010.

[7] B. Efron, T. Hastie, I. Johnstone, and R. Tibshirani. Least angle regression. *The Ann. of Stat.*, 32, 2004.

[8] J. Fan and R. Li. Variable selection via nonconcave penalized likelihood and its oracle properties. *J. of the Am. Stat. Assoc.*, 96(456):1348–1360, 2001.

[9] M. Figueiredo. Adaptive sparseness for supervised learning. *IEEE Trans. on PAMI*, 25(9), 2003.

[10] I. E. Frank and J. H. Friedman. A statistical view of some chemometrics regression tools. *Technometrics*, 35(2):109–135, 1993.

[11] J. Friedman, T. Hastie, and R. Tibshirani. Sparse inverse covariance estimation with the graphical lasso. *Biostatistics*, 9(3), 2008.

[12] D. Heckerman, C. Meek, and G. F. Cooper. A Bayesian approach to causal discovery. In C. Glymour and G. F. Cooper, editors, *Computation, Causation, and Discovery*. MIT, 1999.

[13] G. J. Huang, S. Shifman, and W. Valdar et. al. High resolution mapping of expression QTLs in heterogeneous stock mice in multiple tissues. *Genome Research*, 19(6):1133–40, 2009.

[14] J. Jia and B. Yu. On model selection consistency of the elastic net when $p \gg n$. Technical Report 756, UC Berkeley, Department of Statistics, 2008.

[15] M. B. Katan. Apolipoprotein E isoforms, serum cholesterol and cancer. *Lancet*, i:507–508, 1986.

[16] S. Kim and E. Xing. Statistical estimation of correlated genome associations to a quantitative trait network. *PLOS Genetics*, 5(8), 2009.

[17] D. A. Lawlor, R. M. Harbord, and J. Sterne et. al. Mendelian randomization: using genes as instruments for making causal inferences in epidemiology. *Stat. in Medicine*, 27:1133–1163, 2008.

[18] E. Levina, A. Rothman, and J. Zhu. Sparse estimation of large covariance matrices via a nested lasso penalty. *The Ann. of App. Stat.*, 2(1):245–263, 2008.

[19] M. H. Maathius, M. Kalisch, and P. Buhlmann. Estimating high-dimensional intervention effects from observation data. *The Ann. of Stat.*, 37:3133–3164, 2009.

[20] D. J. C. MacKay. Bayesian interpolation. *Neural Computation*, 4:415–447, 1992.

[21] D. J. C. MacKay. *Information Theory, Inference & Learning Algorithms*. Cambridge Uni Press, 2003.

[22] J. Mooij, D. Janzing, J. Peters, and B. Schoelkopf. Regression by dependence minimization and its application to causal inference in additive noise models. In *ICML*, 2009.

[23] R. Mott, C. J. Talbot, M. G. Turri, A. C. Collins, and J. Flint. A method for fine mapping quantitative trait loci in outbred animal stocks. *Proc. Nat. Acad. Sci. USA*, 97:12649–12654, 2000.

[24] C. J. Narvaez and D. Matthews et. al. Lean phenotype and resistance to diet-induced obesity in vitamin D receptor knockout mice correlates with induction of uncoupling protein-1. *Endocrinology*, 150(2), 2009.

[25] R. M. Neal. *Bayesian Learning for Neural Networks*. Springer, 1996.

[26] T. Park and G. Casella. The Bayesian LASSO. *J. of the Am. Stat. Assoc.*, 103(482), 2008.

[27] J. Pearl. *Causality: Models, Reasoning, and Inference*. Cambridge Uni Press, 2000.

[28] J. Pearl. Causal inference in statistics: an overview. *Statistics Surveys*, 3:96–146, 2009.

[29] J. M. Robins and S. Greenland. Identification of causal effects using instrumental variables: comment. *J. of the Am. Stat. Assoc.*, 91:456–458, 1996.

[30] E. E. Schadt, J. Lamb, X. Yang, and J. Zhu et. al. An integrative genomics approach to infer causal associations between gene expression and disease. *Nature Genetics*, 37(7):710–717, 2005.

[31] M. W. Seeger. Bayesian inference and optimal design for the sparse linear model. *JMLR*, 9, 2008.

[32] I. Shpitser and J. Pearl. Identification of conditional interventional distributions. In *UAI*, 2006.

[33] R. Silva, R. Scheines, C. Glymour, and P. Spirtes. Learning the structure of linear latent variable models. *JMLR*, 7, 2006.

[34] G. D. Smith and S. Ebrahim. Mendelian randomisation: can genetic epidemiology contribute to understanding environmental determinants of disease? *Int. J. of Epidemiology*, 32:1–22, 2003.

[35] D.C. Thomas and D.V. Conti. Commentary: The concept of Mendelian randomization. *Int. J. of Epidemiology*, 32, 2004.

[36] R. Tibshirani. Regression shrinkage and selection via the lasso. *JRSS B*, 58(1):267–288, 1996.

[37] M. E. Tipping. Sparse Bayesian learning and the RVM. *JMLR*, 1:211–244, 2001.

[38] W. Valdar, L. C. Solberg, and S. Burnett et. al. Genome-wide genetic association of complex traits in heterogeneous stock mice. *Nature Genetics*, 38:879–887, 2006.

[39] M. Wainwright. Sharp thresholds for high-dimensional and noisy sparsity recovery using L1-constrained quadratic programmming. *IEEE Trans. on Inf. Theory*, 55:2183 – 2202, 2007.

[40] M. Yuan and Y. Lin. On the nonnegative garrote estimator. *JRSS:B*, 69, 2007.

[41] J. Zhu, M. C. Wiener, and C. Zhang et. al. Increasing the power to detect causal associations by combining genotypic and expression data in segregating populations. *PLOS Comp. Biol.*, 3(4):692–703, 2007.

[42] H. Zou and T. Hastie. Regularization and variable selection via the elastic net. *JRSS:B*, 67(2), 2005.

